# Semiparametric Support Vector and Linear Programming Machines

**Alex J. Smola, Thilo T. Frieß, and Bernhard Schölkopf**
GMD FIRST, Rudower Chaussee 5, 12489 Berlin
{smola, friess, bs}@first.gmd.de

## Abstract

Semiparametric models are useful tools in the case where domain knowledge exists about the function to be estimated or emphasis is put onto understandability of the model. We extend two learning algorithms - Support Vector machines and Linear Programming machines to this case and give experimental results for SV machines.

## 1 Introduction

One of the strengths of Support Vector (SV) machines is that they are *nonparametric* techniques, where one does not have to e.g. specify the number of basis functions beforehand. In fact, for many of the kernels used (not the polynomial kernels) like Gaussian rbf–kernels it can be shown [6] that SV machines are universal approximators.

While this is advantageous in general, parametric models are useful techniques in their own right. Especially if one happens to have additional knowledge about the problem, it would be unwise not to take advantage of it. For instance it might be the case that the major properties of the data are described by a combination of a small set of linear independent basis functions $\{\phi_1(\cdot), \ldots, \phi_n(\cdot)\}$. Or one may want to correct the data for some (e.g. linear) trends. Secondly it also may be the case that the user wants to have an *understandable* model, without sacrificing accuracy. For instance many people in life sciences tend to have a preference for linear models. This may be some motivation to construct *semiparametric* models, which are both easy to understand (for the parametric part) and perform well (often due to the nonparametric term). For more advocacy on semiparametric models see [1].

A common approach is to fit the data with the parametric model and train the nonparametric add–on on the errors of the parametric part, i.e. fit the nonparametric part to the errors. We show in Sec. 4 that this is useful only in a very restricted

situation. In general it is impossible to find the best model amongst a given class for different cost functions by doing so. The better way is to solve a convex optimization problem like in standard SV machines, however with a different set of admissible functions

$$f(x) = \langle w, \psi(x) \rangle + \sum_{i=1}^{n} \beta_i \phi_i(x). \tag{1}$$

Note that this is not so much different from the classical SV [10] setting where one uses functions of the type

$$f(x) = \langle w, \psi(x) \rangle + b. \tag{2}$$

## 2  Semiparametric Support Vector Machines

Let us now treat this setting more formally. For the sake of simplicity in the exposition we will restrict ourselves to the case of SV regression and only deal with the $\varepsilon$–insensitive loss function $|\xi|_\varepsilon = \max\{0, |\xi| - \varepsilon\}$. Extensions of this setting are straightforward and follow the lines of [7].

Given a training set of size $\ell$, $X := \{(x_1, y_1), \ldots, (x_\ell, y_\ell)\}$ one tries to find a function $f$ that minimizes the functional of the expected risk[1]

$$R[f] = \int c(f(x) - y) p(x, y) dx dy. \tag{3}$$

Here $c(\xi)$ denotes a cost function, i.e. how much deviations between prediction and actual training data should be penalized. Unless stated otherwise we will use $c(\xi) = |\xi|_\varepsilon$.

As we do not know $p(x, y)$ we can only compute the empirical risk $R_{\text{emp}}[f]$ (i.e. the training error). Yet, minimizing the latter is not a good idea if the model class is sufficiently rich and will lead to overfitting. Hence one adds a regularization term $T[f]$ and minimzes the regularized risk functional

$$R_{\text{reg}}[f] = \sum_{i=1}^{\ell} c(f(x_i) - y_i) + \lambda T[f] \text{ with } \lambda > 0. \tag{4}$$

The standard choice in SV regression is to set $T[f] = \frac{1}{2}\|w\|^2$.

This is the point of departure from the standard SV approach. While in the latter $f$ is described by (2), we will expand $f$ in terms of (1). Effectively this means that there exist functions $\phi_1(\cdot), \ldots, \phi_n(\cdot)$ whose contribution is not regularized at all. If $n$ is sufficiently smaller than $\ell$ this need not be a major concern, as the VC–dimension of this additional class of linear models is $n$, hence the overall capacity control will still work, provided the nonparametric part is restricted sufficiently. Figure 1 explains the effect of choosing a different structure in detail.

Solving the optimization equations for this particular choice of a regularization term, with expansion (1), the $\varepsilon$–insensitive loss function and introducing kernels

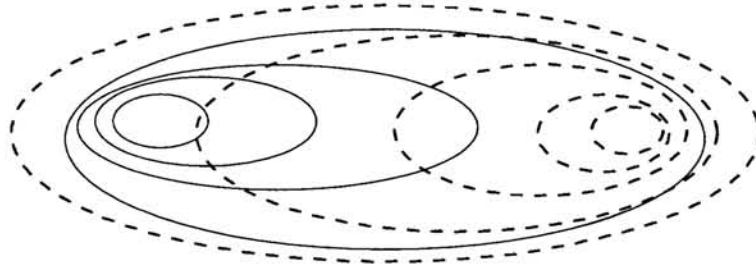

Figure 1: Two different nested subsets (solid and dotted lines) of hypotheses and the optimal model (+) in the realizable case. Observe that the optimal model is already contained in much a smaller (in this diagram size corresponds to the capacity of a subset) subset of the structure with solid lines than in the structure denoted by the dotted lines. Hence prior knowledge in choosing the structure can have a large effect on generalization bounds and performance.

following [2] we arrive at the following primal optimization problem:

$$\text{minimize} \quad \tfrac{\lambda}{2}\|w\|^2 + \sum_{i=1}^{\ell} \xi_i + \xi_i^*$$

$$\text{subject to} \quad \begin{cases} \langle w, \psi(x_i)\rangle + \sum_{j=1}^{n} \beta_j \phi_j(x_i) - y_i &\leq\ \epsilon + \xi_i^* \\ y_i - \langle w, \psi(x_i)\rangle - \sum_{j=1}^{n} \beta_j \phi_j(x_i) &\leq\ \epsilon + \xi_i \\ \xi_i, \xi_i^* &\geq\ 0 \end{cases} \qquad (5)$$

Here $k(x, x')$ has been written as $\langle \psi(x), \psi(x')\rangle$. Solving (5) for its Wolfe dual yields

$$\text{maximize} \quad \begin{cases} -\tfrac{1}{2} \sum_{i,j=1}^{\ell} (\alpha_i - \alpha_i^*)(\alpha_j - \alpha_j^*)k(x_i, x_j) \\ -\varepsilon \sum_{i=1}^{\ell}(\alpha_i + \alpha_i^*) + \sum_{i=1}^{\ell} y_i(\alpha_i - \alpha_i^*) \end{cases}$$

$$(6)$$

$$\text{subject to} \quad \begin{cases} \sum_{i=1}^{\ell}(\alpha_i - \alpha_i^*)\phi_j(x_i) &=\ 0 \text{ for all } 1 \leq j \leq n \\ \alpha_i, \alpha_i^* &\in\ [0, 1/\lambda] \end{cases}$$

Note the similarity to the standard SV regression model. The objective function and the box constraints on the Lagrange multipliers $\alpha_i, \alpha_i^*$ remain unchanged. The only modification comes from the additional unregularized basis functions. Whereas in the standard SV case we only had a single (constant) function $b \cdot 1$ we now have an expansion in the basis $\beta_i \phi_i(\cdot)$. This gives rise to $n$ constraints instead of one. Finally $f$ can be found as

$$f(x) = \sum_{i=1}^{\ell}(\alpha_i - \alpha_i^*)k(x_i, x) + \sum_{i=1}^{n} \beta_i \phi_i(x) \quad \text{since} \quad w = \sum_{i=1}^{\ell}(\alpha_i - \alpha_i^*)\psi(x_i). \quad (7)$$

The only difficulty remaining is how to determine $\beta_i$. This can be done by exploiting the Karush–Kuhn–Tucker optimality conditions, or much more easily, by using an interior point optimization code [9]. In the latter case the variables $\beta_i$ can be obtained as the dual variables of the dual (dual dual = primal) optimization problem (6) as a by product of the optimization process. This is also how these variables have been obtained in the experiments in the current paper.

# 3   Semiparametric Linear Programming Machines

Equation (4) gives rise to the question whether not completely different choices of regularization functionals would also lead to good algorithms. Again we will allow functions as described in (7). Possible choices are

$$T[f] = \frac{1}{2}\|w\|^2 + \sum_{i=1}^{n} |\beta_i| \tag{8}$$

or

$$T[f] = \sum_{i=1}^{\ell} |\alpha_i - \alpha_i^*| \tag{9}$$

or

$$T[f] = \sum_{i=1}^{\ell} |\alpha_i - \alpha_i^*| + \frac{1}{2} \sum_{i,j=1}^{n} \beta_i \beta_j M_{ij} \tag{10}$$

for some positive semidefinite matrix $M$. This is a simple extension of existing methods like Basis Pursuit [3] or Linear Programming Machines for classification (see e.g. [4]). The basic idea in all these approaches is to have two different sets of basis functions that are regularized differently, or where a subset may not be regularized at all. This is an efficient way of encoding prior knowledge or the preference of the user as the emphasis obviously will be put mainly on the functions with little or no regularization at all. Eq. (8) is essentially the SV estimation model where an additional linear regularization term has been added for the parametric part. In this case the constraints of the optimization problem (6) change into

$$
\begin{aligned}
-1 \;\leq\; & \sum_{i=1}^{\ell}(\alpha_i - \alpha_i^*)\phi_j(x_i) \;\leq\; 1 && \text{for all } 1 \leq j \leq n \\
& \alpha_i, \alpha_i^* \qquad\qquad\quad \in\; [0, 1/\lambda]
\end{aligned}
\tag{11}
$$

It makes little sense (from a technical viewpoint) to compute Wolfe's dual objective function in (10) as the problem does not get significantly easier by doing so. The best approach is to solve the corresponding optimization problem directly by some linear or quadratic programming code, e.g. [9]. Finally (10) can be reduced to the case of (8) by renaming variables accordingly and a proper choice of $M$.

# 4   Why Backfitting is not sufficient

One might think that the approach presented above is quite unnecessary and overly complicated for semiparametric modelling. In fact, one could try to fit the data to the parametric model first, and then fit the nonparametric part to the residuals. In most cases, however, this does not lead to finding the minimum of (4). We will show this at a simple example.

Take a SV machine with linear kernel (i.e. $k(x, x') = \langle x, x' \rangle$) in one dimension and a constant term as parametric part (i.e. $f(x) = wx + \beta$). This is one of the simplest semiparametric SV machines possible. Now suppose the data was generated by

$$y_i = x_i \quad \text{where} \quad x_i \geq 1 \tag{12}$$

without noise. Clearly then also $y_i \geq 1$ for all $i$. By construction the best overall fit of the pair $(\beta, w)$ will be arbitrarily close to $(0, 1)$ if the regularization parameter $\lambda$ is chosen sufficiently small.

For backfitting one first carries out the parametric fit to find a constant $\beta$ minimizing the term $\sum_{i=1}^{\ell} c(y_i - \beta)$. Depending on the chosen cost function $c(\cdot)$, $\beta$ will be the mean ($L_2$–error), the median ($L_1$–error), etc., of the set $\{y_1, \dots, y_\ell\}$. As all $y_i \geq 1$

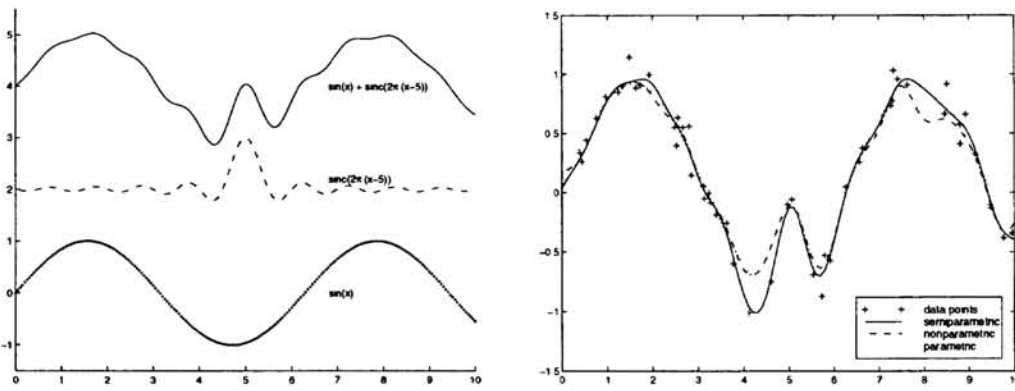

Figure 2: Left: Basis functions used in the toy example. Note the different length scales of $\sin x$ and $\mathrm{sinc}\, 2\pi x$. For convenience the functions were shifted by an offset of 2 and 4 respectively. Right: Training data denoted by '+', nonparametric (dash–dotted line), semiparametric (solid line), and parametric regression (dots). The regularization constant was set to $\lambda = 2$. Observe that the semiparametric model picks up the characteristic *wiggles* of the original function.

also $\beta \geq 1$ which is surely not the optimal solution of the overall problem as there $\beta$ would be close to 0 as seen above. Hence not even in the simplest of all settings backfitting minimizes the regularized risk functional, thus one cannot expect the latter to happen in the more complex case either. There exists only one case in which backfitting would suffice, namely if the function spaces spanned by the kernel expansion $\{k(x_i, \cdot)\}$ and $\{\phi_i(\cdot)\}$ were orthogonal. Consequently in general one has to jointly solve for both the parametric and the semiparametric part.

## 5 Experiments

The main goal of the experiments shown is a proof of concept and to display the properties of the new algorithm. We study a modification of the *Mexican hat* function, namely

$$f(x) = \sin x + \mathrm{sinc}(2\pi(x - 5)). \tag{13}$$

Data is generated by an additive noise process, i.e. $y_i = f(x_i) + \xi_i$, where $\xi_i$ is additive noise. For the experiments we choose Gaussian rbf–kernels with width $\sigma = 1/4$, normalized to maximum output 1. The noise is uniform with 0.2 standard deviation, the $\varepsilon$–insensitive cost function $|\cdot|_\varepsilon$ with $\varepsilon = 0.05$. Unless stated otherwise averaging is done over 100 datasets with 50 samples each. The $x_i$ are drawn uniformly from the interval $[0, 10]$. $L_1$ and $L_2$ errors are computed on the interval $[0, 10]$ with uniform measure. Figure 2 shows the function and typical predictions in the nonparametric, semiparametric, and parametric setting. One can observe that the semiparametric model including $\sin x$, $\cos x$ and the constant function as basis functions generalizes better than the standard SV machine. Fig. 3 shows that the generalization performance is better in the semiparametric case. The length of the weight vector of the kernel expansion $\|w\|$ is displayed in Fig. 4. It is smaller in the semiparametric case for practical values of the regularization strength. To make a more realistic comparison, model selection (how to determine $1/\lambda$) was carried out by 10–fold cross validation for both algorithms independently for all 100 datasets. Table 1 shows generalization performance for both a nonparametric model, a correctly chosen and an incorrectly chosen semiparametric model. The experiments indicate that cases in which prior knowledge exists on the type of functions to be used will benefit from semiparametric modelling. Future experiments will show how much can be gained in real world examples.

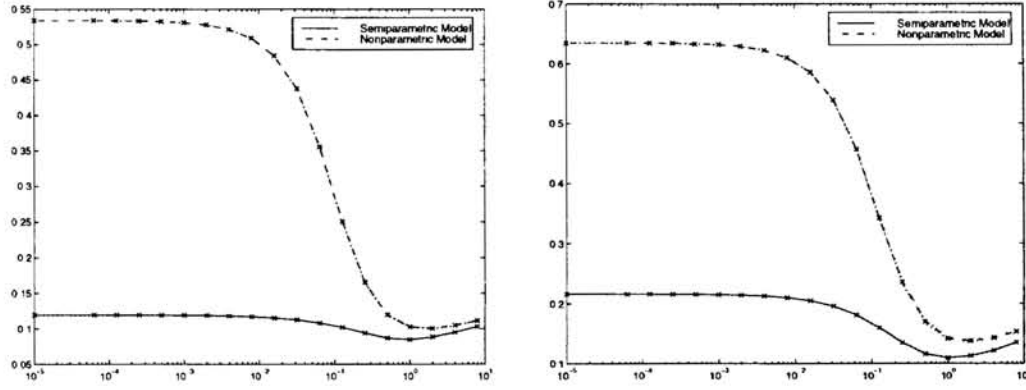

Figure 3: $L_1$ error (left) and $L_2$ error (right) of the nonparametric / semiparametric regression computed on the interval $[0, 10]$ vs. the regularization strength $1/\lambda$. The dotted lines (although hardly visible) denote the variance of the estimate. Note that in both error measures the semiparametric model consistently outperforms the nonparametric one.

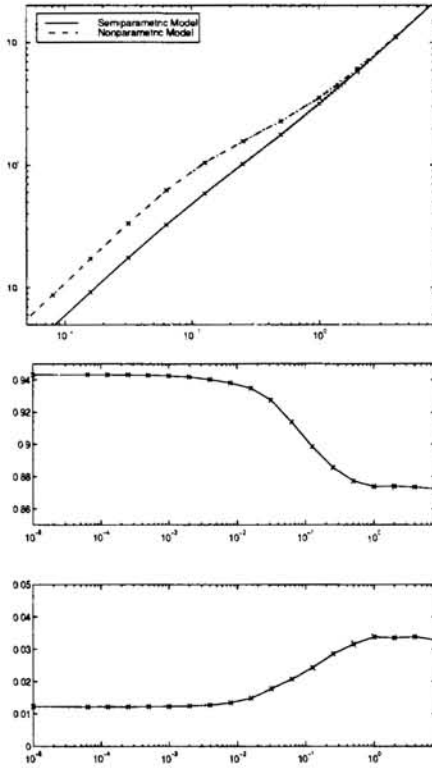

Figure 4: Length of the weight vector $w$ in *feature space* $(\sum_{i,j}(\alpha_i - \alpha_i^*)(\alpha_j - \alpha_j^*)k(x_i, x_j))^{1/2}$ vs. regularization strength. Note that $\|w\|$, controlling the capacity of that part of the function, belonging to the kernel expansion, is smaller (for practical choices of the regularization term) in the semiparametric than in the nonparametric model. If this difference is sufficiently large the overall capacity of the resulting model is smaller in the semiparametric approach. As before dotted lines indicates the variance.

Figure 5: Estimate of the parameters for $\sin x$ (top picture) and $\cos x$ (bottom picture) in the semiparametric model vs. regularization strength $1/\lambda$. The dotted lines above and below show the variation of the estimate given by its variance. Training set size was $\ell = 50$. Note the small variation of the estimate. Also note that even in the parametric case $1/\lambda \approx 0$ neither the coefficient for $\sin x$ converges to 1, nor does the corresponding term for $\cos x$ converge to 0. This is due to the additional frequency contributions of sinc $2\pi x$.

|  | Nonparam. | Semiparam. $\sin x, \cos x, 1$ | Semiparam. $\sin 2x, \cos 2x, 1$ |
|---|---|---|---|
| $L_1$ error | $0.1263 \pm 0.0064$ (12) | $0.0887 \pm 0.0018$ (82) | $0.1267 \pm 0.0064$ (6) |
| $L_2$ error | $0.1760 \pm 0.0097$ (12) | $0.1197 \pm 0.0046$ (82) | $0.1864 \pm 0.0124$ (6) |

Table 1: $L_1$ and $L_2$ error for model selection by 10–fold crossvalidation. The *correct* semiparametric model ($\sin x, \cos x, 1$) outperforms the nonparametric model by at least 30%, and has significantly smaller variance. The wrongly chosen nonparametric model ($\sin 2x, \cos 2x, 1$), on the other hand, gives performance comparable to the nonparametric one, in fact, no significant performance degradation was noticeable. The number in parentheses denotes the number of trials in which the corresponding model was the best among the three models.

## 6 Discussion and Outlook

Similar models have been proposed and explored in the context of smoothing splines. In fact, expansion (7) is a direct result of the representer theorem, however only in the case of regularization in feature space (aka Reproducing Kernel Hilbert Space, RKHS). One can show [5] that the expansion (7) is optimal in the space spanned by the RKHS and the additional set of basis functions.

Moreover the semiparametric setting arises naturally in the context of conditionally positive definite kernels of order $m$ (see [8]). There, in order to use a set of kernels which do not satisfy Mercer's condition, one has to exclude polynomials up to order $m - 1$. Hence, to with that one has to add polynomials back in 'manually' and our approach presents a way of doing that.

Another application of semiparametric models besides the conventional approach of treating the nonparametric part as *nuisance parameters* [1] is the domain of hypothesis testing, e.g. to test whether a parametric model fits the data sufficiently well. This can be achieved in the framework of structural risk minimization [10] — given the different models (nonparametric vs. semiparametric vs. parametric) one can evaluate the bounds on the expected risk and then choose the model with the lowest error bound. Future work will tackle the problem of computing good error bounds of compound hypothesis classes. Moreover it should be easily possible to apply the methods proposed in this paper to Gaussian processes.

**Acknowledgements** This work was supported in part by grants of the DFG Ja 379/51 and ESPRIT Project Nr. 25387-STORM. The authors thank Peter Bartlett, Klaus–Robert Müller, Noboru Murata, Takashi Onoda, and Bob Williamson for helpful discussions and comments.

## Footnotes

[1]More general definitions, mainly in terms of the cost function, do exist but for the sake of clarity in the exposition we ignored these cases. See [10] or [7] for further details on alternative definitions of risk functionals.

## References

[1] P.J. Bickel, C.A.J. Klaassen, Y. Ritov, and J.A. Wellner. *Efficient and adaptive estimation for semiparametric models.* J. Hopkins Press, Baltimore, ML, 1994.

[2] B. E. Boser, I. M. Guyon, and V. N. Vapnik. A training algorithm for optimal margin classifiers. In *COLT'92*, pages 144–152, Pittsburgh, PA, 1992.

[3] S. Chen, D. Donoho, and M. Saunders. Atomic decomposition by basis pursuit. Technical Report 479, Department of Statistics, Stanford University, 1995.

[4] T.T. Frieß and R.F. Harrison. Perceptrons in kernei feature spaces. TR RR-720, University of Sheffield, Sheffield, UK, 1998.

[5] G.S. Kimeldorf and G. Wahba. A correspondence between Bayesan estimation on stochastic processes and smoothing by splines. *Ann. Math. Statist.*, 2:495–502, 1971.

[6] C.A. Micchelli. Interpolation of scattered data: distance matrices and conditionally positive definite functions. *Constructive Approximation*, 2:11–22, 1986.

[7] A. J. Smola and B. Schölkopf. On a kernel–based method for pattern recognition, regression, approximation and operator inversion. *Algorithmica*, 22:211–231, 1998.

[8] A.J. Smola, B. Schölkopf, and K. Müller. The connection between regularization operators and support vector kernels. *Neural Netw.*, 11:637–649, 1998.

[9] R.J. Vanderbei. LOQO: An interior point code for quadratic programming. TR SOR-94-15, Statistics and Operations Research, Princeton Univ., NJ, 1994.

[10] V. Vapnik. *The Nature of Statistical Learning Theory.* Springer, N.Y., 1995.